# Graphical Models via Generalized Linear Models

**Eunho Yang**
Department of Computer Science
University of Texas at Austin
eunho@cs.utexas.edu

**Pradeep Ravikumar**
Department of Computer Science
University of Texas at Austin
pradeepr@cs.utexas.edu

**Genevera I. Allen**
Department of Statistics
Rice University
gallen@rice.edu

**Zhandong Liu**
Department of Pediatrics-Neurology
Baylor College of Medicine
zhandonl@bcm.edu

## Abstract

Undirected graphical models, also known as Markov networks, enjoy popularity in a variety of applications. The popular instances of these models such as Gaussian Markov Random Fields (GMRFs), Ising models, and multinomial discrete models, however do not capture the characteristics of data in many settings. We introduce a new class of graphical models based on generalized linear models (GLMs) by assuming that node-wise conditional distributions arise from exponential families. Our models allow one to estimate multivariate Markov networks given any univariate exponential distribution, such as Poisson, negative binomial, and exponential, by fitting penalized GLMs to select the neighborhood for each node. A major contribution of this paper is the rigorous statistical analysis showing that with high probability, the neighborhood of our graphical models can be recovered exactly. We also provide examples of non-Gaussian high-throughput genomic networks learned via our GLM graphical models.

## 1 Introduction

Undirected graphical models, also known as Markov random fields, are an important class of statistical models that have been extensively used in a wide variety of domains, including statistical physics, natural language processing, image analysis, and medicine. The key idea in this class of models is to represent the joint distribution as a product of clique-wise compatibility functions; given an underlying graph, each of these compatibility functions depends only on a subset of variables within any clique of the underlying graph. Such a factored graphical model distribution can also be related to an exponential family distribution [1], where the unnormalized probability is expressed as the exponential of a weighted linear combination of clique-wise sufficient statistics. Learning a graphical model distribution from data within this exponential family framework can be reduced to learning weights on these sufficient statistics. An important modeling question is then, how do we choose suitable sufficient statistics? In the case of discrete random variables, sufficient statistics can be taken as indicator functions as in the Ising or Potts model. These, however, are not suited to all kinds of discrete variables such as that of non-negative integer counts. Similarly, in the case of continuous variables, Gaussian Markov Random Fields (GMRFs) are popular. The multivariate normal distribution imposed by the GMRF, however, is a stringent assumption; the marginal distribution of any variable must also be Gaussian.

In this paper, we propose a general class of graphical models beyond the Ising model and the GMRF to encompass variables arising from all exponential family distributions. Our approach is motivated by recent state of the art methods for learning the standard Ising and Gaussian MRFs [2, 3, 4].

The key idea in these recent methods is to learn the MRF graph structure by estimating node-neighborhoods, which are estimated by maximizing the likelihood of each node conditioned on the rest of the nodes. These node-wise fitting methods have been shown to be both computationally and statistically attractive. Here, we study the general class of models obtained by the following construction: suppose the node-conditional distributions of each node conditioned on the rest of the nodes are Generalized Linear Models (GLMs) [5]. By the Hammersley-Clifford Theorem [6] and some algebra as derived in [7], these node-conditional distributions entail a global distribution that factors according to cliques defined by the graph obtained from the node-neighborhoods. Moreover, these have a particular set of potential functions specified by the GLM. The resulting class of MRFs broadens the class of models available off-the-shelf, from the standard Ising, indicator-discrete, and Gaussian MRFs.

Beyond our initial motivation of finding more general graphical model sufficient statistics, a broader class of parametric graphical models are important for a number of reasons. First, our models provide a principled approach to model multivariate distributions and network structures among a large number of variables. For many non-Gaussian exponential families, multivariate distributions typically do not exist in an analytical or computationally tractable form. Graphical model GLMs provide a way to "extend" univariate exponential families of distributions to the multivariate case and model and study relationships between variables for these families of distributions. Second, while some have proposed to extend the GMRF to a non-parametric class of graphical models by first Gaussianizing the data and then fitting a GMRF over the transformed variables [8], the sample complexity of such non-parametric methods is often inferior to parametric methods. Thus for modeling data that closely follows a non-Gaussian distribution, statistical power for network recovery can be gained by directly fitting parametric GLM graphical models. Third, and specifically for multivariate count data, others have suggested combinatorial approaches to fitting graphical models, mostly in the context of contingency tables [6, 9, 1, 10]. These approaches, however, are computationally intractable for even moderate numbers of variables.

Finally, potential applications for our GLM graphical models abound. Networks of call-times, time spent on websites, diffusion processes, and life-cycles can be modeled with exponential graphical models; other skewed multivariate data can be modeled with gamma or chi-squared graphical models. Perhaps the most interesting motivating applications are for multivariate count data such as from website visits, user-ratings, crime and disease incident reports, bibliometrics, and next-generation genomic sequencing technologies. The latter is a relatively new high-throughput technology to measure gene expression that is rapidly replacing the microarray [11]. As Gaussian graphical models are widely used to infer genomic regulatory networks from microarray data, Poisson and negative binomial graphical models may be important for inferring genomic networks from the multivariate count data arising from this emerging technology. Beyond next generation sequencing, there has been a recent proliferation of new high-throughput genomic technologies that produce non-Gaussian data. Thus, our more general class of GLM graphical models can be used for inferring genomic networks from these new high-throughput technologies.

The construction of our GLM graphical models also suggests a natural method for learning such models: node-wise neighborhood estimation by fitting sparsity constrained GLMs. A main contribution of this paper is to provide a sparsistency analysis for the recovery of the underlying graph structure of this new class of MRFs. The presence of non-linearities arising from the GLM poses subtle technical issues not present in the linear case [2]. Indeed, for the specific cases of logistic, and multinomial respectively, [3, 4] derive such a sparsistency analysis via fairly extensive arguments which were tuned to those specific cases. Here, we generalize their analysis to general GLMs, which requires a slightly modified M-estimator and a more subtle theoretical analysis. We note that this analysis might be of independent interest even outside the context of modeling and recovering graphical models. In recent years, there has been a trend towards unified statistical analyses that provide statistical guarantees for broad classes of models via general theorems [12]. Our result is in this vein and provides structure recovery for the class of sparsity constrained generalized linear models. We hope that the techniques we introduce might be of use to address the outstanding question of sparsity constrained M-estimation in its full generality.

## 2 A New Class of Graphical Models

***Problem Setup and Background.*** Suppose $X = (X_1, \ldots, X_p)$ is a random vector, with each variable $X_i$ taking values in a set $\mathcal{X}$. Suppose $G = (V, E)$ is an undirected graph over $p$ nodes corresponding to the $p$ variables; the corresponding graphical model is a set of distributions that satisfy Markov independence assumptions with respect to the graph. By the Hammersley-Clifford theorem, any such distribution also factors according to the graph in the following way. Let $\mathcal{C}$ be a set of cliques (fully-connected subgraphs) of the graph $G$, and let $\{\phi_c(X_c) \ c \in \mathcal{C}\}$ be a set of clique-wise sufficient statistics. With this notation, any distribution of $X$ within the graphical model family represented by the graph $G$ takes the form:

$$P(X) \quad \propto \quad \exp\left\{\sum_{c \in \mathcal{C}} \theta_c \phi_c(X_c)\right\}, \tag{1}$$

where $\{\theta_c\}$ are weights over the sufficient statistics. With a pairwise graphical model distribution, the set of cliques consists of the set of nodes $V$ and the set of edges $E$, so that

$$P(X) \quad \propto \quad \exp\left\{\sum_{s \in V} \theta_s \phi_s(X_s) + \sum_{(s,t) \in E} \theta_{st} \phi_{st}(X_s, X_t)\right\}. \tag{2}$$

As previously discussed, an important question is how to select the class of sufficient statistics, $\phi$, in particular to obtain as a multivariate extension of specified univariate parametric distributions? We next outline a subclass of graphical models where the node-conditional distributions are exponential family distributions, with an important special case where these node-conditional distributions are generalized linear models (GLMs). Then, in Section 3, we will study how to learn the underlying graph structure, or infer the edge set $E$, providing an M-estimator and sufficient conditions under which the estimator recovers the graph structure with high probability.

***Graphical Models via GLMs.*** In this section, we investigate the class of models that arise from specifying the node-conditional distributions as exponential families. Specifically, suppose we are given a univariate exponential family distribution,

$$P(Z) = \exp(\theta\, B(Z) + C(Z) - D(\theta)),$$

with sufficient statistics $B(Z)$, base measure $C(Z)$, and $D(\theta)$ as the log-normalization constant.

Let $X = (X_1, X_2, \ldots, X_p)$ be a $p$-dimensional random vector; and let $G = (V, E)$ be an undirected graph over $p$ nodes corresponding to the $p$ variables. Now suppose the distribution of $X_s$ given the rest of nodes $X_{V \setminus s}$ is given by the above exponential family, but with the canonical exponential family parameter set to a linear combination of $k$-th order products of univariate functions $\{B(X_t)\}_{t \in N(s)}$. This gives the following conditional distribution:

$$P(X_s | X_{V \setminus s}) = \exp\left\{B(X_s)\left(\theta_s + \sum_{t \in N(s)} \theta_{st}\, B(X_t) + \sum_{t_2, t_3 \in N(s)} \theta_{s\, t_2 t_3}\, B(X_{t_2}) B(X_{t_3})\right.\right.$$

$$\left.\left. + \sum_{t_2, \ldots, t_k \in N(s)} \theta_{s\, t_2 \ldots t_k} \prod_{j=2}^{k} B(X_{t_j})\right) + C(X_s) - \bar{D}(X_{V \setminus s})\right\}, \tag{3}$$

where $C(X_s)$ is specified by the exponential family, and $\bar{D}(X_{V \setminus s})$ is the log-normalization constant.

By the Hammersley-Clifford theorem, and some elementary calculation, this conditional distribution can be shown to specify the following unique joint distribution $P(X_1, \ldots, X_p)$:

**Proposition 1.** *Suppose $X = (X_1, X_2, \ldots, X_p)$ is a p-dimensional random vector, and its node-conditional distributions are specified by (3). Then its joint distribution $P(X_1, \ldots, X_p)$ is given by:*

$$P(X) = \exp\left\{\sum_s \theta_s B(X_s) + \sum_{s \in V} \sum_{t \in N(s)} \theta_{st}\, B(X_s) B(X_t)\right.$$

$$\left. + \sum_{s \in V} \sum_{t_2, \ldots, t_k \in N(s)} \theta_{s \ldots t_k}\, B(X_s) \prod_{j=2}^{k} B(X_{t_j}) + \sum_s C(X_s) - A(\theta)\right\}, \tag{4}$$

*where $A(\theta)$ is the log-normalization constant.*

An important question is whether the conditional and joint distributions specified above have the most general form, under just the assumption of exponential family node-conditional distributions? In particular, note that the canonical parameter in the previous proposition is a *tensor factorization* of the univariate sufficient statistic, with pair-wise and higher-order interactions, which seems a bit stringent. Interestingly, by extending the argument from [7] and the Hammersley-Clifford Theorem, we can show that indeed (3) and (4) have the most general form.

**Proposition 2.** *Suppose $X = (X_1, X_2, \ldots, X_p)$ is a $p$-dimensional random vector, and its node-conditional distributions are specified by an exponential family,*

$$P(X_s | X_{V \setminus s}) = \exp\{E(X_{V \setminus s}) \, B(X_s) + C(X_s) - \bar{D}(X_{V \setminus s})\}, \qquad (5)$$

*where the function $E(X_{V \setminus s})$ (and hence the log-normalization constant $\bar{D}(X_{V \setminus s})$) only depends on variables $X_t$ in $N(s)$. Further, suppose the corresponding joint distribution factors according to the graph $G = (V, E)$, with the factors over cliques of size at most $k$. Then, the conditional distribution in (5) has the tensor-factorized form in (3), and the corresponding joint distribution has the form in (4).*

The proposition thus tells us that under the general assumptions that (a) the joint distribution is a graphical model that factors according to a graph $G$, and has clique-factors of size at most $k$, and (c) its node-conditional distribution follows an exponential family, it *necessarily* follows that the conditional and joint distributions are given by (3) and (4) respectively.

An important special case is when the joint distribution has factors of size at most two. The conditional distribution then is given by:

$$P(X_s | X_{V \setminus s}) = \exp\left\{ \theta_s \, B(X_s) + \sum_{t \in N(s)} \theta_{st} \, B(X_s) B(X_t) + C(X_s) - \bar{D}(X_{V \setminus s}) \right\}, \qquad (6)$$

while the joint distribution is given as

$$P(X) = \exp\left\{ \sum_s \theta_s B(X_s) + \sum_{(s,t) \in E} \theta_{st} \, B(X_s) B(X_t) + \sum_s C(X_s) - A(\theta) \right\}. \qquad (7)$$

Note that when the univariate sufficient statistic function $B(\cdot)$ is a linear function $B(X_s) = X_s$, then the conditional distribution in (6) is precisely a generalized linear model [5] in canonical form,

$$P(X_s | X_{V \setminus s}) = \exp\left\{ \theta_s \, X_s + \sum_{t \in N(s)} \theta_{st} \, X_s \, X_t + C(X_s) - \bar{D}(X_{V \setminus s}; \theta) \right\}, \qquad (8)$$

while the joint distribution has the form,

$$P(X) = \exp\left\{ \sum_s \theta_s X_s + \sum_{(s,t) \in E} \theta_{st} \, X_s \, X_t + \sum_s C(X_s) - A(\theta) \right\}. \qquad (9)$$

In the subsequent sections, we will refer to the entire class of models in (7) as GLM graphical models, but focus on the case (9) with linear functions $B(X_s) = X_s$.

***Examples***. The GLM graphical models provide multivariate or Markov network extensions of univariate exponential family distributions. The popular Gaussian graphical model and Ising model can thus also be represented by (7). Consider the latter, for example, where for the Bernoulli distribution, we have that $B(X) = X, C(X) = 0$, and $A(\theta)$ is the log-partition function; plugging these into (9), we have the form of the Ising model studied in [3]. The form of the multinomial graphical model, an extension of the Ising model, can also be represented by (7) and has been previously studied in [4] and others.

It is instructive to consider the domain of the set of all possible valid parameters in the GLM graphical model (9); namely those that ensure that the density is normalizable, or equivalently, so that the log-partition function satisfies $A(\theta) < +\infty$. The Ising model imposes no constraint on its parameters, $\{\theta_{st}\}$, for normalizability, since there are finitely many configurations of the binary random

vector $X$. For other exponential families, with countable discrete or continuous valued variables, the GLM graphical model does impose additional constraints on valid parameters. Consider the example of the Poisson and exponential distributions. The Poisson family has sufficient statistic $B(X) = X$ and base measure $C(X) = -\log(X!)$. With some algebra, we can show that $A(\theta) < +\infty$ implies $\theta_{st} \leq 0 \; \forall \, s, t$. Thus, the Poisson graphical model can only capture negative conditional relationships between variables. Consider the exponential distribution with sufficient statistic $B(X) = -X$, base measure $C(X) = 0$. To ensure that the density is finitely integrable, so that $A(\theta) < +\infty$, we then require that $\theta_{st} \geq 0 \; \forall \, s, t$. Similar constraints on the parameter space are necessary to ensure proper density functions for several other exponential family graphical models as well.

## 3 Statistical Guarantees

In this section, we study the problem of learning the graph structure of an underlying GLM graphical model given iid samples. Specifically, we assume that we are given $n$ samples $X_1^n = \{X^{(i)}\}_{i=1}^n$, from a GLM graphical model:

$$P(X; \theta^*) = \exp \left\{ \sum_{(s,t) \in E^*} \theta_{st}^* \, X_s \, X_t + \sum_s C(X_s) - A(\theta) \right\}. \tag{10}$$

We have removed node-wise terms for simplicity, noting that our analysis extends to the general case. The goal in graphical model structure recovery is to recover the edges $E^*$ of the underlying graph $G = (V, E^*)$. Following [3, 4], we will approach this problem via neighborhood estimation, where we estimate the neighborhood of each node individually, and then stitch these together to form the global graph estimate. Specifically, if we have an estimate $\widehat{\mathcal{N}}(s)$ for the true neighborhood $\mathcal{N}^*(s)$, then we can estimate the overall graph structure as:

$$\widehat{E} = \cup_{s \in V} \cup_{t \in \widehat{\mathcal{N}}(s)} \{(s,t)\}. \tag{11}$$

In order to estimate the neighborhood of any node, we consider the sparsity constrained conditional MLE. Given the joint distribution in (10), the conditional distribution of $X_s$ given the rest of the nodes is given by:

$$P(X_s | X_{V \setminus s}) = \exp \left\{ X_s \Big( \sum_{t \in N(s)} \theta_{st}^* X_t \Big) + C(X_s) - D \Big( \sum_{t \in N(s)} \theta_{st}^* X_t \Big) \right\}. \tag{12}$$

Let $\theta_{\setminus s}^* = \{\theta_{st}^*\}_{t \in V \setminus s} \in \mathbb{R}^{p-1}$ be a zero-padded vector, with entries $\theta_{st}^*$ for $t \in N(s)$ and $\theta_{st}^* = 0$, for $t \notin N(s)$. Given $n$ samples $X_1^n = \{X^{(i)}\}_{i=1}^n$, we can write the conditional log-likelihood of the distribution (12) as:

$$\ell(\theta_{\setminus s}; X_1^n) := -\frac{1}{n} \log \prod_{i=1}^n P\big(X_s^{(i)} | X_{\setminus s}^{(i)}, \theta_{\setminus s}\big) = \frac{1}{n} \sum_{i=1}^n -X_s^{(i)} \langle \theta_{\setminus s}, X_{\setminus s}^{(i)} \rangle + D\big(\langle \theta_{\setminus s}, X_{\setminus s}^{(i)} \rangle\big).$$

We can then solve the $\ell_1$ regularized conditional log-likelihood loss for each node $X_s$:

$$\min_{\theta_{\setminus s} \in \mathbb{R}^{p-1}} \ell(\theta_{\setminus s}; X_1^n) + \lambda_n \|\theta_{\setminus s}\|_1. \tag{13}$$

Given the solution $\widehat{\theta}_{\setminus s}$ of the M-estimation problem above, we then estimate the node-neighborhood of $s$ as $\widehat{N}(s) = \{t \in V \setminus s : \widehat{\theta}_{st} \neq 0\}$. In the following when we focus on a fixed node $s \in V$, we will overload notation, and use $\theta \in \mathbb{R}^{p-1}$ as the parameters of the conditional distribution, suppressing the dependence on $s$.

In the rest of the section, we first discuss the assumptions we impose on the GLM graphical model parameters. The first set of assumptions are standard irrepresentable-type conditions imposed for structure recovery in high-dimensional statistical estimators, and in particular, our assumptions mirror those in [3]. The second set of assumptions are key to our generalized analysis of the class of GLM graphical models as a whole. We then follow with our main theorem, that guarantees structure recovery under these assumptions, with high probability even in high-dimensional regimes.

Our first set of assumptions use the Fisher Information matrix, $Q_s^* = \nabla^2 \ell(\theta_s^*; X_1^n)$, which is the Hessian of the node-conditional log-likelihood. In the following, we will simply use $Q^*$ instead of $Q_s^*$ where the reference node $s$ should be understood implicitly. We also use $S = \{(s,t) : t \in N(s)\}$ to denote the true neighborhood of node $s$, and $S^c$ to denote its complement. We use $Q_{SS}^*$ to denote the $d \times d$ sub-matrix indexed by $S$. Our first two assumptions , and are as follows:

**Assumption 1** (Dependency condition)**.** There exists a constant $\lambda_{\min} > 0$ such that $\lambda_{\min}(Q_{SS}^*) \geq \lambda_{\min}$. Moreover, there exists a constant $\lambda_{\max} < \infty$ such that $\lambda_{\max}(\widehat{E}[X_{\backslash s} X_{\backslash s}^T]) \leq \lambda_{\max}$.

**Assumption 2** (Incoherence condition)**.** We also need an incoherence or irrepresentable condition on the fisher information matrix as in [3]. Specifically, there exists a constant $\alpha > 0$, such that $\max_{t \in S^c} \|Q_{tS}^*(Q_{SS}^*)^{-1}\|_1 \leq 1 - \alpha$.

A key technical facet of the linear, logistic, and multinomial models in [2, 3, 4] and used heavily in their proofs, is that the random variables $\{X_s\}$ there were bounded with high probability. Unfortunately, in the general GLM distribution in (12), we cannot assume this explicitly. Nonetheless, we show that we can analyze the corresponding regularized M-estimation problems, provided the first and second moments are bounded.

**Assumption 3.** The first and second moments of the distribution in (10) are bounded as follows. The first moment $\mu^* := \mathbb{E}[X]$, satisfies $\|\mu^*\|_2 \leq \kappa_m$; the second moment satisfies $\max_{t \in V} \mathbb{E}[X_t^2] \leq \kappa_v$.

We also need smoothness assumptions on the log-normalization constants :

**Assumption 4.** The log-normalization constant $A(\cdot)$ of the joint distribution (10) satisfies: $\max_{u : \|u\| \leq 1} \lambda_{\max}(\nabla^2 A(\theta^* + u)) \leq \kappa_h$.

**Assumption 5.** The log-partition function $D(\cdot)$ of the node-conditional distribution (12) satisfies: There exist constants $\kappa_1$ and $\kappa_2$ (that depend on the exponential family) s.t. $\max\{|D''(\kappa_1 \log \eta)|, |D'''(\kappa_1 \log \eta)|\} \leq n^{\kappa_2}$ where $\eta = \max\{n, p\}$, $\kappa_1 \geq \frac{9}{2}\|\theta^*\|_2$ and $\kappa_2 \in [0, 1/4]$.

Assumptions 3 and 4 are the key technical conditions under which we can generalize the analyses in [2, 3, 4] to the general GLM case. In particular, we can show that the statements of the following propositions hold, which show that the random vectors $X$ following the GLM graphical model in (10) are suitably well-behaved:

**Proposition 3.** *Suppose $X$ is a random vector with the distribution specified in* (10)*. Then, for any vector $u \in \mathbb{R}^p$ such that $\|u\|_2 \leq c'$, any positive constant $\delta$, and some constants $c > 0$,*

$$P\big(|\langle u, X \rangle| \geq \delta \log \eta\big) \leq c\eta^{-\delta/c'}.$$

**Proposition 4.** *Suppose $X$ is a random vector with the distribution specified in* (10)*. Then, for $\delta \leq \min\{2\kappa_v/3, \kappa_h + \kappa_v\}$, and some constant $c > 0$,*

$$P\left(\frac{1}{n}\sum_{i=1}^n \big(X_s^{(i)}\big)^2 \geq \delta\right) \leq 2\exp\big(-c\,n\,\delta^2\big).$$

Putting these key technical results and assumptions together, we arrive at our main result:

**Theorem 1.** *Consider a GLM graphical model distribution as specified in* (10)*, with true parameter $\theta^*$ and associated edge set $E^*$ that satisfies Assumptions 1-5. Suppose that $\min_{(s,t)\in E^*} |\theta_{st}^*| \geq \frac{10}{\lambda_{\min}}\sqrt{d}\lambda_n$ where $d$ is the maximum neighborhood size. Suppose also that the regularization parameter is chosen such that $\lambda_n \geq M\frac{(2-\alpha)}{\alpha}\sqrt{\frac{\log p}{n^{1-\kappa_2}}}$ for some constant $M > 0$. Then, there exist positive constants $L$, $K_1$ and $K_2$ such that if $n \geq L\left\{d^2 \log p(\max\{\log n, \log p\})^2\right\}^{\frac{1}{1-3\kappa_2}}$, then with probability at least $1 - \exp(-K_1\lambda_n^2 n) - K_2 \max\{n, p\}^{-5/4}$, the following statements hold:*

(a) *(Unique Solution) For each node $s \in V$, the solution of the M-estimation problem in* (13) *is unique, and*

(b) *(Correct Neighborhood Recovery) The M-estimate also recovers the true neighborhood exactly, so that $\widehat{N}(s) = N(s)$.*

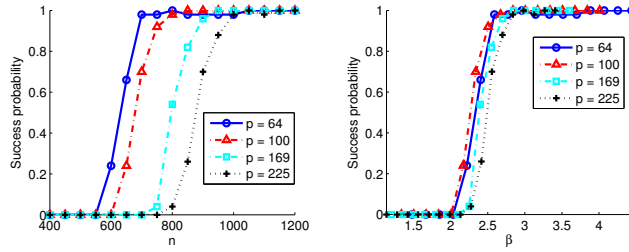

Figure 1: Probabilities of successful support recovery for a Poisson grid structure ($\omega = -0.1$). The probability of successful edge recovery vs. n (Left), and the probability of successful edge recovery vs. control parameter $\beta = n/(c \log p)$ (Right).

Note that if the neighborhood of each node is recovered with high probability, then by a simple union bound, the estimate in (11), $\widehat{E} = \cup_{s \in V} \cup_{t \in \widehat{\mathcal{N}}(s)} \{(s,t)\}$ is equal to the true edge set $E^*$ with high-probability.

Also note that $\kappa_2$ in the statement is a constant from Assumption 5. The Poisson family has one of the steepest log-partition function: $D(\eta) = \exp(\eta)$. Hence, in order to satisfy Assumption 5, we need $\|\theta^*\|_2 \leq \frac{1}{18} \frac{\log n}{\log p}$ with $\kappa_2 = 1/4$. On the other hand, for the binomial, multinomial or Gaussian cases studied in [2, 3, 4], we can recover their results with $\kappa_2 = 0$ since the log-partition function $D(\cdot)$ of these families are upper bounded by some constant for any input. Nevertheless, we need to restrict $\theta^*$ to satisfy Assumption 4 so that the variables are bounded with high probability in Proposition 3 and 4 for any GLM case.

## 4 Experiments

***Experiments on Simulated Networks***. We provide a small simulation study that demonstrates the consequences of Theorem 1 when the conditional distribution in (12) has the form of Poisson distribution. We performed experiments on lattice (4 nearest neighbor) graphs with identical edge weight $\omega$ for all edges. Simulating data via Gibbs sampling, we solved the sparsity-constrained optimization problem with a constant factor of $\sqrt{\frac{\log p}{n}}$ for $\lambda_n$. The left panel of Figure 1 shows the probability of successful edge recovery for different numbers of nodes, $p = \{64, 100, 169, 225\}$. In the right panel of Figure 1, we re-scale the sample size $n$ using the "control parameter" $\beta = n/(c \log p)$ for some constant $c$. Each point in the plot indicates the probability that all edges are successfully recovered out of 50 trials. We can see that the curves for different problem sizes are well aligned with the results of Theorem 1.

***Learning Genomic Networks***. Gaussian graphical models learned from microarray data have often been used to study high-throughput genomic regulatory networks. Our GLM graphical models will be important for understanding genomic networks learned from other high-throughput technologies that do not produce approximately Gaussian data. Here, we demonstrate the versatility of our model by learning two cancer genomic networks, a genomic copy number aberration network (from aCGH data) for Glioblastoma learned by multinomial graphical models and a meta-miRNA inhibitory network (from next generation sequencing data) for breast cancer learned by Poisson graphical models. Level III data, breast cancer miRNA expression (next generation sequencing) [13] and copy number variation (aCGH) Glioblastoma data [14], was obtained from the the Cancer Genome Atlas (TCGA) data portal (http://tcga-data.nci.nih.gov/tcga/), and processed according to standard techniques. Data descriptions and processing details are given in the supplemental materials.

A Poisson graphical model and a multinomial graphical model were fit to the processed miRNA data and aberration data respectively by performing neighborhood selection with the sparsity of the graph determined by stability selection [15]. Our GLM graphical models, Figure 2, reveal results consistent with the cancer genomics literature. The meta-miRNA inhibitory network has three major hubs, two of which, mir-519 and mir-520, are known to be breast cancer tumor suppressors [16, 17]. Interestingly, let-7, a well-known miRNA involved in tumor metastasis [18], plays a central role

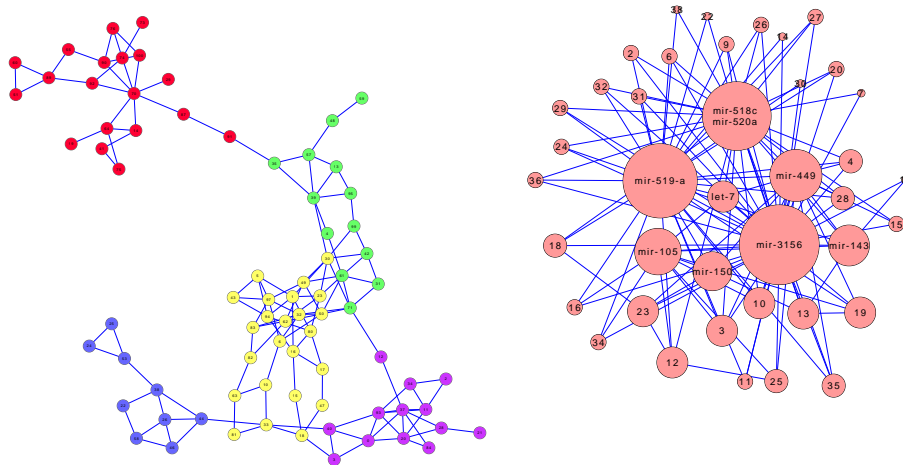

Figure 2: Genomic copy number aberration network for Glioblastoma learned via multinomial graphical models (left) and meta-miRNA inhibitory network for breast cancer learned via Poisson graphical models (right).

in our network, sharing edges with the five largest hubs; this suggests that our model has learned relevant negative associations between tumor suppressors and enhancers. The Glioblastoma copy number aberration network reveals five major modules, color coded on the left panel in Figure 2, and three of these modules have been previously implicated in Glioblastoma: EGFR in the yellow module, PTEN in the purple module, and CDK2A in the blue module [19].

# 5   Discussion

We have introduced a new class of graphical models that arise when we assume that node-wise conditional distributions follow an exponential family distribution. We have also provided simple M-estimators for learning the network by fitting node-wise penalized GLMs that enjoy strong statistical recovery properties. Our work has broadened the class of off-the-shelf graphical models to encompass a wide range of parametric distributions. These classes of graphical models may be of further interest to the statistical community as they provide closed form multivariate densities for several exponential family distributions (e.g. Poisson, exponential, negative binomial) where few currently exist. Furthermore, the statistical analysis of our M-estimator required subtle techniques that may be of general interest in the analysis of sparse M-estimation.

Our work outlines the general class of graphical models for exponential family distributions, but there are many avenues for future work in studying this model for specific distributional families. In particular, our model sometimes places restrictions on the parameter space. A question remains, can these restrictions be relaxed for specific exponential family distributions? Additionally, we have focused on families with linear sufficient statistics (e.g. Gaussian, Bernoulli, Poisson, exponential, negative binomial); our models can be studied with non-linear sufficient statistics or multi-parameter distributions as well. Overall, our work has opened the door for learning Markov Networks from a broad class of distributions, the properties and applications of which leave much room for future research.

**Acknowledgments**

E.Y. and P.R. acknowledge support from NSF IIS-1149803. G.A. and Z.L. acknowledge support from the Collaborative Advances in Biomedical Computing seed funding program at the Ken Kennedy Institute for Information Technology at Rice University supported by the John and Ann Doerr Fund for Computational Biomedicine and by the Center for Computational and Integrative Biomedical Research seed funding program at Baylor College of Medicine. G.A. also acknowledges support from NSF DMS-1209017.

# References

[1] M.J. Wainwright and M.I. Jordan. Graphical models, exponential families, and variational inference. *Foundations and Trends® in Machine Learning*, 1(1-2):1–305, 2008.

[2] N. Meinshausen and P. Bühlmann. High-dimensional graphs and variable selection with the Lasso. *Annals of Statistics*, 34:1436–1462, 2006.

[3] P. Ravikumar, M. J. Wainwright, and J. Lafferty. High-dimensional ising model selection using $\ell_1$-regularized logistic regression. *Annals of Statistics*, 38(3):1287–1319, 2010.

[4] A. Jalali, P. Ravikumar, V. Vasuki, and S. Sanghavi. On learning discrete graphical models using group-sparse regularization. In *Inter. Conf. on AI and Statistics (AISTATS)*, 14, 2011.

[5] P. McCullagh and J.A. Nelder. *Generalized linear models*. Monographs on statistics and applied probability 37. Chapman and Hall/CRC, New York, 1989.

[6] S.L. Lauritzen. *Graphical models*, volume 17. Oxford University Press, USA, 1996.

[7] J. Besag. Spatial interaction and the statistical analysis of lattice systems. *Journal of the Royal Statistical Society. Series B (Methodological)*, 36(2):192–236, 1974.

[8] H. Liu, J. Lafferty, and L. Wasserman. The nonparanormal: Semiparametric estimation of high dimensional undirected graphs. *The Journal of Machine Learning Research*, 10:2295–2328, 2009.

[9] Y.M.M. Bishop, S.E. Fienberg, and P.W. Holland. *Discrete multivariate analysis*. Springer Verlag, 2007.

[10] Trevor. Hastie, Robert. Tibshirani, and JH (Jerome H.) Friedman. *The elements of statistical learning*. Springer, 2 edition, 2009.

[11] J.C. Marioni, C.E. Mason, S.M. Mane, M. Stephens, and Y. Gilad. Rna-seq: an assessment of technical reproducibility and comparison with gene expression arrays. *Genome research*, 18(9):1509–1517, 2008.

[12] S. Negahban, P. Ravikumar, M. J. Wainwright, and B. Yu. A unified framework for high-dimensional analysis of $m$-estimators with decomposable regularizers, 2010.

[13] Cancer Genome Atlas Research Network. Comprehensive molecular portraits of human breast tumours. *Nature*, 490(7418):61–70, 2012.

[14] Cancer Genome Atlas Research Network. Comprehensive genomic characterization defines human glioblastoma genes and core pathways. *Nature*, 455(7216):1061–1068, October 2008.

[15] H. Liu, K. Roeder, and L. Wasserman. Stability approach to regularization selection (stars) for high dimensional graphical models. *Arxiv preprint arXiv:1006.3316*, 2010.

[16] K. Abdelmohsen, M.M. Kim, S. Srikantan, E.M. Mercken, S.E. Brennan, G.M. Wilson, R. de Cabo, and M. Gorospe. mir-519 suppresses tumor growth by reducing hur levels. *Cell cycle (Georgetown, Tex.)*, 9(7):1354, 2010.

[17] I. Keklikoglou, C. Koerner, C. Schmidt, JD Zhang, D. Heckmann, A. Shavinskaya, H. Allgayer, B. Gückel, T. Fehm, A. Schneeweiss, et al. Microrna-520/373 family functions as a tumor suppressor in estrogen receptor negative breast cancer by targeting nf-$\kappa$b and tgf-$\beta$ signaling pathways. *Oncogene*, 2011.

[18] F. Yu, H. Yao, P. Zhu, X. Zhang, Q. Pan, C. Gong, Y. Huang, X. Hu, F. Su, J. Lieberman, et al. let-7 regulates self renewal and tumorigenicity of breast cancer cells. *Cell*, 131(6):1109–1123, 2007.

[19] R. McLendon, A. Friedman, D. Bigner, E.G. Van Meir, D.J. Brat, G.M. Mastrogianakis, J.J. Olson, T. Mikkelsen, N. Lehman, K. Aldape, et al. Comprehensive genomic characterization defines human glioblastoma genes and core pathways. *Nature*, 455(7216):1061–1068, 2008.

[20] Jianhua Zhang. *Convert segment data into a region by sample matrix to allow for other high level computational analyses*, version 1.2.0 edition. Bioconductor package.

[21] Gerald B W Wertheim, Thomas W Yang, Tien-chi Pan, Anna Ramne, Zhandong Liu, Heather P Gardner, Katherine D Dugan, Petra Kristel, Bas Kreike, Marc J van de Vijver, Robert D Cardiff, Carol Reynolds, and Lewis A Chodosh. The Snf1-related kinase, Hunk, is essential for mammary tumor metastasis. *Proceedings of the National Academy of Sciences of the United States of America*, 106(37):15855–15860, September 2009.

[22] J.T. Leek, R.B. Scharpf, H.C. Bravo, D. Simcha, B. Langmead, W.E. Johnson, D. Geman, K. Baggerly, and R.A. Irizarry. Tackling the widespread and critical impact of batch effects in high-throughput data. *Nature Reviews Genetics*, 11(10):733–739, 2010.

[23] J. Li, D.M. Witten, I.M. Johnstone, and R. Tibshirani. Normalization, testing, and false discovery rate estimation for rna-sequencing data. *Biostatistics*, 2011.

[24] G. I. Allen and Z. Liu. A Log-Linear Graphical Model for Inferring Genetic Networks from High-Throughput Sequencing Data. *IEEE International Conference on Bioinformatics and Biomedicine*, 2012.

[25] J. Bullard, E. Purdom, K. Hansen, and S. Dudoit. Evaluation of statistical methods for normalization and differential expression in mrna-seq experiments. *BMC bioinformatics*, 11(1):94, 2010.

